# The IM Algorithm : A variational approach to Information Maximization

**David Barber**      **Felix Agakov**
Institute for Adaptive and Neural Computation : `www.anc.ed.ac.uk`
Edinburgh University, EH1 2QL, U.K.

### Abstract

The maximisation of information transmission over noisy channels is a common, albeit generally computationally difficult problem. We approach the difficulty of computing the mutual information for noisy channels by using a variational approximation. The resulting IM algorithm is analagous to the EM algorithm, yet maximises mutual information, as opposed to likelihood. We apply the method to several practical examples, including linear compression, population encoding and CDMA.

## 1    Introduction

The reliable communication of information over noisy channels is a widespread issue, ranging from the construction of good error-correcting codes to feature extraction[3, 12]. In a neural context, maximal information transmission has been extensively studied and proposed as a principal goal of sensory processing[2, 5, 7]. The central quantity in this context is the Mutual Information (MI) which, for source variables (inputs) $\mathbf{x}$ and response variables (outputs) $\mathbf{y}$, is

$$I(\mathbf{x}, \mathbf{y}) \equiv H(\mathbf{y}) - H(\mathbf{y}|\mathbf{x}), \tag{1}$$

where $H(\mathbf{y}) \equiv -\langle \log p(\mathbf{y}) \rangle_{p(\mathbf{y})}$ and $H(\mathbf{y}|\mathbf{x}) \equiv -\langle \log p(\mathbf{y}|\mathbf{x}) \rangle_{p(\mathbf{x},\mathbf{y})}$ are marginal and conditional entropies respectively, and angled brackets represent averages. The goal is to adjust parameters of the mapping $p(\mathbf{y}|\mathbf{x})$ to maximise $I(\mathbf{x}, \mathbf{y})$. Despite the simplicity of the statement, the MI is generally intractable for all but special cases. The key difficulty lies in the computation of the entropy of $p(\mathbf{y})$ (a mixture).

One such tractable special case is if the mapping $\mathbf{y} = g(\mathbf{x}; \boldsymbol{\Theta})$ is deterministic and invertible, for which the difficult entropy term trivially becomes

$$H(\mathbf{y}) = \langle \log |\mathsf{J}| \rangle_{p(\mathbf{y})} + const. \tag{2}$$

Here $\mathsf{J} = \{\partial y_i / \partial x_j\}$ is the Jacobian of the mapping. For non-Gaussian sources $p(\mathbf{x})$, and special choices of $g(\mathbf{x}; \boldsymbol{\Theta})$, the minimization of (1) with respect to the parameters $\boldsymbol{\Theta}$ leads to the infomax formulation of ICA[4].

Another tractable special case is if the source distribution $p(\mathbf{x})$ is Gaussian and the mapping $p(\mathbf{y}|\mathbf{x})$ is Gaussian.

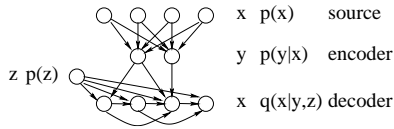

x  p(x)       source

y  p(y|x)    encoder

z p(z)

x  q(x|y,z) decoder

Figure 1: An illustration of the form of a more general mixture decoder. $x$ represents the sources or inputs, which are (stochastically) encoded as $y$. A receiver decodes $y$ (possibly with the aid of auxiliary variables $z$).

However, in general, approximations of the MI need to be considered. A variety of methods have been proposed. In neural coding, a popular alternative is to maximise the Fisher 'Information'[5]. Other approaches use different objective criteria, such as average reconstruction error.

## 2  Variational Lower Bound on Mutual Information

Since the MI is a measure of information transmission, our central aim is to maximise a lower bound on the MI. Using the symmetric property of the MI, an equivalent formulation of the MI is $I(\mathbf{x}, \mathbf{y}) = H(\mathbf{x}) - H(\mathbf{x}|\mathbf{y})$. Since we shall generally be interested in optimising MI with respect to the parameters of $p(\mathbf{y}|\mathbf{x})$, and $p(\mathbf{x})$ is simply the data distribution, we need to bound $H(\mathbf{x}|\mathbf{y})$ suitably. The Kullback-Leibler bound $\sum_{\mathbf{x}} p(\mathbf{x}|\mathbf{y}) \log p(\mathbf{x}|\mathbf{y}) - p(\mathbf{x}|\mathbf{y}) \log q(\mathbf{x}|\mathbf{y}) \geq 0$ gives

$$I(\mathbf{x}, \mathbf{y}) \geq \underbrace{H(\mathbf{x})}_{\text{``}entropy\text{''}} + \underbrace{\langle \log q(\mathbf{x}|\mathbf{y}) \rangle_{p(\mathbf{x}, \mathbf{y})}}_{\text{``}energy\text{''}} \stackrel{\text{def}}{=} \tilde{I}(\mathbf{x}, \mathbf{y}). \tag{3}$$

where $q(\mathbf{x}|\mathbf{y})$ is an arbitrary variational distribution. The bound is exact if $q(\mathbf{x}|\mathbf{y}) \equiv p(\mathbf{x}|\mathbf{y})$. The form of this bound is convenient since it explicitly includes both the encoder $p(\mathbf{y}|\mathbf{x})$ and decoder $q(\mathbf{x}|\mathbf{y})$, see fig(1).

Certainly other well known lower bounds on the MI may be considered [6] and a future comparison of these different approaches would be interesting. However, our current experience suggests that the bound considered above is particularly computationally convenient. Since the bound is based on the KL divergence, it is equivalent to a moment matching approximation of $p(\mathbf{x}|\mathbf{y})$ by $q(\mathbf{x}|\mathbf{y})$. This fact is highly beneficial in terms of decoding, since mode matching approaches, such as mean-field theory, typically get trapped in the one of many sub-optimal local minima. More successful decoding algorithms approximate the posterior mean[10].

**The IM algorithm**

To maximise the MI with respect to any parameters $\theta$ of $p(\mathbf{y}|\mathbf{x}, \theta)$, we aim to push up the lower bound (3). First one needs to choose a class of variational distributions $q(\mathbf{x}|\mathbf{y}) \in Q$ for which the energy term is tractable. Then a natural recursive procedure for maximising $\tilde{I}(X, Y)$ for given $p(\mathbf{x})$, is

1. For fixed $q(\mathbf{x}|\mathbf{y})$, find $\theta^{new} = \arg\max_{\theta} \tilde{I}(X, Y)$

2. For fixed $\theta$, $q^{new}(\mathbf{x}|\mathbf{y}) = \arg\max_{q(\mathbf{x}|\mathbf{y}) \in Q} \tilde{I}(X, Y)$, where $Q$ is a chosen class of distributions.

These steps are iterated until convergence. This procedure is analogous to the (G)EM algorithm which maximises a lower bound on the likelihood[9]. The difference is simply in the form of the "energy" term.

Note that if $|\mathbf{y}|$ is large, the posterior $p(\mathbf{x}|\mathbf{y})$ will typically be sharply peaked around its mode. This would motivate a simple approximation $q(\mathbf{x}|\mathbf{y})$ to the posterior,

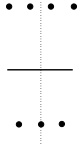

Figure 2: The MI optimal linear projection of data $\mathbf{x}$ (dots) is not always given by PCA. PCA projects data onto the vertical line, for which the entropy conditional on the projection $H(\mathbf{x}|\mathbf{y})$ is large. Optimally, we should project onto the horizontal line, for which the conditional entropy is zero.

significantly reducing the computational complexity of optimization. In the case of real-valued $\mathbf{x}$, a natural choice in the large $|\mathbf{y}|$ limit is to use a Gaussian. A simple approximation would then be to use a Laplace approximation to $p(\mathbf{x}|\mathbf{y})$ with covariance elements $[\Sigma^{-1}]_{ij} = \frac{\partial^2 \log p(\mathbf{x}|\mathbf{y})}{\partial x_i \partial x_j}$. Inserted in the bound, this then gives a form reminiscent of the Fisher Information[5]. The bound presented here is arguably more general and appropriate than presented in [5] since, whilst it also tends to the exact value of the MI in the limit of a large number of responses, it is a principled bound for any response dimension.

### Relation to Conditional Likelihood

Consider an autoencoder $x \to y \to \tilde{x}$ and imagine that we wish to maximise the probability that the reconstruction $\tilde{x}$ is in the same $s$ state as $x$:

$$\log p(\tilde{x} = s|x = s) = \log \int_y p(\tilde{x} = s|y)p(y|x = s) \overset{Jensen}{\geq} \langle \log p(\tilde{x} = s|y) \rangle_{p(y|x=s)}$$

Averaging this over all the states of $x$:

$$\sum_s p(x = s) \log p(\tilde{x} = s|x = s) \geq \sum_s \langle \log p(\tilde{x} = s|y) \rangle_{p(x=s,y)} \equiv \langle \log q(x|y) \rangle_{p(x,y)}$$

Hence, maximising $\tilde{I}(X, Y)$ (for fixed $p(x)$) is the same as maximising the lower bound on the probability of a correct reconstruction. This is a reassuring property of the lower bound. Even though we do not directly maximise the MI, we also indirectly maximise the probability of a correct reconstruction – a form of autoencoder.

### Generalisation to Mixture Decoders

A straightforward application of Jensen's inequality leads to the more general result:

$$I(X, Y) \geq H(X) + \langle \log q(x|y, z) \rangle_{p(y|x)p(x)q(z)} \equiv \tilde{I}(X, Y)$$

where $q(x|y, z)$ and $q(z)$ are variational distributions. The aim is to choose $q(x|y, z)$ such that the bound is tractably computable. The structure is illustrated in fig(1).

## 3   Linear Gaussian Channel : Improving on PCA

A common theme in linear compression and feature extraction is to map a (high dimensional) vector $\mathbf{x}$ to a (lower dimensional) vector $\mathbf{y} = W\mathbf{x}$ such that the information in the vector $\mathbf{x}$ is maximally preserved in $\mathbf{y}$. The classical solution to this problem (and minimizes the linear reconstruction error) is given by PCA. However, as demonstrated in fig(2), the optimal setting for $W$ is, in general *not* given by the widely used PCA.

To see how we might improve on the PCA approach, we consider optimising our bound with respect to linear mappings. We take as our projection (encoder) model,

$p(\mathbf{y}|\mathbf{x}) \sim \mathcal{N}(\mathbf{Wx}, s^2\mathbf{I})$, with isotropic Gaussian noise. The empirical distribution is simply $p(\mathbf{x}) \propto \sum_{\mu=1}^{P} \delta(\mathbf{x} - \mathbf{x}^\mu)$, where $P$ is the number of datapoints. Without loss of generality, we assume the data is zero mean. For a decoder $q(\mathbf{x}|\mathbf{y}) = \mathcal{N}(\mathbf{m}(\mathbf{y}), \Sigma(\mathbf{y}))$, maximising the bound on MI is equivalent to minimising

$$\sum_{\mu=1}^{P} \left\langle (\mathbf{x} - \mathbf{m}(\mathbf{y}))^T \Sigma^{-1}(\mathbf{y})(\mathbf{x} - \mathbf{m}(\mathbf{y})) + \log \det \Sigma(\mathbf{y}) \right\rangle_{p(\mathbf{y}|\mathbf{x}^\mu)}$$

For constant diagonal matrices $\Sigma(\mathbf{y})$, this reduces to minimal mean square reconstruction error autoencoder training in the limit $s^2 \to 0$. This clarifies why autoencoders (and hence PCA) are a sub-optimal special case of MI maximisation.

### Linear Gaussian Decoder

A simple decoder is given by $q(\mathbf{x}|\mathbf{y}) \sim \mathcal{N}(\mathbf{Uy}, \sigma^2\mathbf{I})$, for which

$$\tilde{I}(\mathbf{x}, \mathbf{y}) \propto 2\mathrm{tr}(\mathbf{UWS}) - \mathrm{tr}(\mathbf{UMU}^T), \quad (4)$$

where $\mathbf{S} = \langle \mathbf{xx}^T \rangle = \sum_\mu \mathbf{x}^\mu(\mathbf{x}^\mu)^T/P$ is the sample covariance of the data, and

$$\mathbf{M} = \mathbf{I}s^2 + \mathbf{WSW}^T \quad (5)$$

is the covariance of the mixture distribution $p(\mathbf{y})$. Optimization of (4) for $\mathbf{U}$ leads to $\mathbf{SW}^T = \mathbf{UM}$. Eliminating $\mathbf{U}$, this gives

$$\tilde{I}(\mathbf{x}, \mathbf{y}) \propto \mathrm{tr}\left(\mathbf{SW}^T\mathbf{M}^{-1}\mathbf{WS}\right) \quad (6)$$

In the zero noise limit, optimisation of (6) produces PCA. For noisy channels, unconstrained optimization of (6) leads to a divergence of the matrix norm $\|\mathbf{WW}^T\|_\infty$; a norm-constrained optimisation in general produces a different result to PCA. The simplicity of the linear decoder in this case severely limits any potential improvement over PCA, and certainly would not resolve the issue in fig(2). For this, a non-linear decoder $q(\mathbf{x}|\mathbf{y})$ is required, for which the integrals become more complex.

### Non-linear Encoders and Kernel PCA

An alternative to using non-linear decoders to improve on PCA is to use a non-linear *encoder*. A useful choice is

$$p(\mathbf{y}|\mathbf{x}) = \mathcal{N}(\mathbf{W\Phi}(\mathbf{x}), \sigma^2\mathbf{I})$$

where $\mathbf{\Phi}(\mathbf{x})$ is in general a high dimensional, non-linear embedding function, for which $\mathbf{W}$ will be non-square. In the zero-noise limit the optimal solution for the encoder results in non-linear PCA on the covariance $\langle \mathbf{\Phi}(\mathbf{x})\mathbf{\Phi}(\mathbf{x})^T \rangle$ of the transformed data. By Mercer's theorem, the elements of the covariance matrix may be replaced by a Kernel function of the users choice[8]. An advantage of our framework is that our bound enables the principled comparison of embedding functions/kernels.

## 4 Binary Responses (Neural Coding)

In a neurobiological context, a popular issue is how to encode real-valued stimuli in a population of spiking neurons. Here we look briefly at a simple case in which each neuron fires ($y_i = 1$) with increasing probability the further the membrane potential $\mathbf{w}_i^T\mathbf{x}$ is above threshold $-b_i$. Independent neural firing suggests:

$$p(\mathbf{y}|\mathbf{x}) = \prod_i p(y_i|\mathbf{x}) \stackrel{\text{def}}{=} \prod \sigma(y_i(\mathbf{w}_i^T\mathbf{x} + b_i)). \quad (7)$$

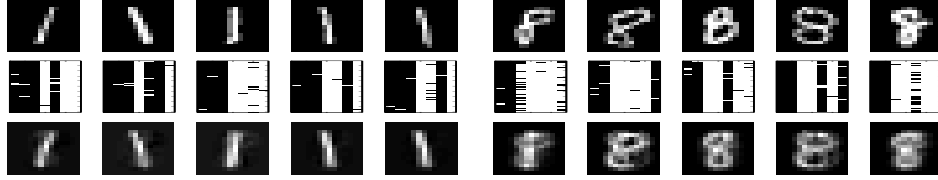

Figure 3: Top row: a subset of the original real-valued source data. Middle row: after training, 20 samples from each of the 7 output units, for each of the corresponding source inputs. Bottom row: Reconstruction of the source data from 50 samples of the output units. Note that while the $8^{th}$ and the $10^{th}$ patterns have closely matching stochastic binary representations, they differ in the firing rates of unit 5. This results in a visibly larger bottom loop of the $8^{th}$ reconstructed pattern, which agrees with the original source data. Also, the thick vertical 1 (pattern 3) differs from the thin vertical eight (pattern 6) due to the differences in stochastic firings of the third and the seventh units.

Here the response variables $\mathbf{y} \in \{-1, +1\}^{|\mathbf{y}|}$, and $\sigma(a) \overset{\text{def}}{=} 1/(1 + e^{-a})$. For the decoder, we chose a simple linear Gaussian $q(\mathbf{x}|\mathbf{y}) \sim \mathcal{N}(\mathbf{Uy}, \mathbf{\Sigma})$. In this case, exact evaluation of the bound (3) is straightforward, since it only involves computations of the second-order moments of $\mathbf{y}$ over the factorized distribution.

A reasonable reconstruction of the source $\mathbf{x}^\star$ from its representation $\mathbf{y}$ will be given by the mean $\tilde{\mathbf{x}} = \langle \mathbf{x} \rangle_{q(\mathbf{x}|\mathbf{y})}$ of the learned approximate posterior. In noisy channels we need to average over multiple possible representations, i.e. $\tilde{\mathbf{x}} = \langle\langle \mathbf{x} \rangle_{q(\mathbf{x}|\mathbf{y})}\rangle_{p(\mathbf{y}|\mathbf{x}^\star)}$.

We performed reconstruction of continuous source data from stochastic binary responses for $|\mathbf{x}| = 196$ input and $|\mathbf{y}| = 7$ output units. The bound was optimized with respect to the parameters of $p(\mathbf{y}|\mathbf{x})$ and $q(\mathbf{x}|\mathbf{y})$ with isotropic norm constraints on $\mathbf{W}$ and $\mathbf{b}$ for 30 instances of digits 1 and 8 (15 of each class). The source variables were reconstructed from 50 samples of the corresponding binary representations at the mean of the learned $q(\mathbf{x}|\mathbf{y})$, see fig(3).

## 5   Code Division Multiple Access (CDMA)

In CDMA[11], a mobile phone user $j \in 1, \ldots, M$ wishes to send a bit $s_j \in \{0, 1\}$ of information to a base station. To send $s_j = 1$, she transmits an $N$ dimensional real-valued vector $\mathbf{g}^j$, which represents a time-discretised waveform ($s_j = 0$ corresponds to no transmission). The simultaneous transmissions from all users results in a received signal at the base station of

$$r_i = \sum_j g_i^j s_j + \eta_i, \qquad i = 1, \ldots, N, \qquad \text{or} \qquad \mathbf{r} = G\mathbf{s} + \boldsymbol{\eta}$$

where $\eta_i$ is Gaussian noise. Probabilistically, we can write

$$p(\mathbf{r}|\mathbf{s}) \propto \exp\left\{ -(\mathbf{r} - G\mathbf{s})^2 /(2\sigma^2) \right\}.$$

The task for the base station (which knows $G$) is to decode the received vector $\mathbf{r}$ so that $\mathbf{s}$ can be recovered reliably. For simplicity, we assume that $N = M$ so that the matrix $G$ is square. Using Bayes' rule, $p(\mathbf{s}|\mathbf{r}) \propto p(\mathbf{r}|\mathbf{s})p(\mathbf{s})$, and assuming a flat prior on $\mathbf{s}$,

$$p(\mathbf{s}|\mathbf{r}) \propto \exp\left\{ -\left(-2\mathbf{r}^T G\mathbf{s} + \mathbf{s}^T G^T G\mathbf{s}\right)/(2\sigma^2)\right\} \tag{8}$$

Computing either the MAP solution $\arg\max_\mathbf{s} p(\mathbf{s}|\mathbf{r})$ or the MPM solution $\arg\max_{s_j} p(s_j|\mathbf{r}), j = 1, \ldots, M$ is, in general, NP-hard.

If $G^T G$ is diagonal, optimal decoding is easy, since the posterior factorises, with

$$p(s_j|\mathbf{r}) \propto \exp\left\{\left(2\sum_i r_i G_{ji} - D_{jj}\right) s_j/(2\sigma^2)\right\}$$

where the diagonal matrix $D = G^T G$ (and we used $s_i^2 \equiv s_i$ for $s_i \in \{0,1\}$). For suitably randomly chosen matrices $G$, $G^T G$ will be approximately diagonal in the limit of large $N$. However, ideally, one would like to construct decoders that perform near-optimal decoding without recourse to the approximate diagonality of $G^T G$. The MAP decoder solves the problem

$$min_{\mathbf{s}\in\{0,1\}^N} \left(\mathbf{s}^T G^T G\mathbf{s} - 2\mathbf{s}^T G^T \mathbf{r}\right) \equiv min_{\mathbf{s}\in\{0,1\}^N} \left(\mathbf{s} - G^{-1}\mathbf{r}\right)^T G^T G \left(\mathbf{s} - G^{-1}\mathbf{r}\right)$$

and hence the MAP solution is that $\mathbf{s}$ which is closest to the vector $G^{-1}\mathbf{r}$. The difficulty lies in the meaning of 'closest' since the space is non-isotropically warped by the matrix $G^T G$. A useful guess for the decoder is that it is the closest in the Euclidean sense to the vector $G^{-1}\mathbf{r}$. This is the so-called decorrelation estimator.

**Computing the Mutual Information**

Of prime interest in CDMA is the evaluation of decoders in the case of non-orthogonal matrices $G$[11]. In this respect, a principled comparison of decoders can be obtained by evaluating the corresponding bound on the MI[1],

$$I(\mathbf{r}, \mathbf{s}) \equiv H(\mathbf{s}) - H(\mathbf{s}|\mathbf{r}) \geq H(\mathbf{s}) + \sum_{\mathbf{r}}\sum_{\mathbf{s}} p(\mathbf{s})p(\mathbf{r}|\mathbf{s}) \log q(\mathbf{s}|\mathbf{r}) \qquad (9)$$

where $H(\mathbf{s})$ is trivially given by $M$ (bits). The bound is exact if $q(\mathbf{s}|\mathbf{r}) = p(\mathbf{s}|\mathbf{r})$.

We make the specific assumption in the following that our decoding algorithm takes the factorised form $q(\mathbf{s}|\mathbf{r}) = \prod_i q(s_i|\mathbf{r})$ and, without loss of generality, we may write

$$q(s_i|\mathbf{r}) = \sigma\left((2s_i - 1)f_i(\mathbf{r})\right) \qquad (10)$$

for some decoding function $f_i(\mathbf{r})$. We restrict interest here to the case of simple linear decoding functions

$$f_i(\mathbf{r}) = a_i + \sum_j w_{ij} r_j.$$

Since $p(\mathbf{r}|\mathbf{s})$ is Gaussian, $(2s_i - 1)f_i(\mathbf{r}) \equiv x_i$ is also Gaussian,

$$p(x_i|\mathbf{s}) = \mathcal{N}(\mu_i(\mathbf{s}), var_i), \qquad \mu_i(\mathbf{s}) \equiv (2s_i - 1)(a_i + \mathbf{w}_i^T G\mathbf{s}), \qquad var_i \equiv \sigma^2 \mathbf{w}_i^T \mathbf{w}_i$$

where $\mathbf{w}_i^T$ is the $i^{th}$ row of the matrix $[W]_{ij} \equiv w_{ij}$. Hence

$$-H(\mathbf{s}|\mathbf{r}) \geq \sum_i \frac{1}{\sqrt{2\pi\sigma^2\mathbf{w}_i^T\mathbf{w}_i}} \left\langle \int_{x=-\infty}^\infty [\log\sigma(x)]\, e^{-[x-(2s_i-1)(a_i+\mathbf{w}_i^T G\mathbf{s})]^2/(2\sigma^2\mathbf{w}_i^T\mathbf{w}_i)} \right\rangle_{p(\mathbf{s})}$$

$$(11)$$

In general, the average over the factorised distribution $p(\mathbf{s})$ can be evaluated by using the Fourier Transform [1]. However, to retain clarity here, we constrain the decoding matrix $W$ so that $\mathbf{w}_i^T G\mathbf{s} = b_i s_i$, i.e. $WG = diag(\mathbf{b})$, for a parameter vector $\mathbf{b}$. The average over $p(\mathbf{s})$ then gives

$$-H(\mathbf{s}|\mathbf{r}) \geq \frac{1}{2} \sum_i \left\langle \log\sigma(x)\left(1 + e^{-[-2xb_i - 4xa_i + 2a_i b_i + b_i^2]/(2\sigma^2\mathbf{w}_i^T\mathbf{w}_i)}\right)\right\rangle_{\mathcal{N}(-a_i, var=\sigma^2\mathbf{w}_i^T\mathbf{w}_i)},$$

$$(12)$$

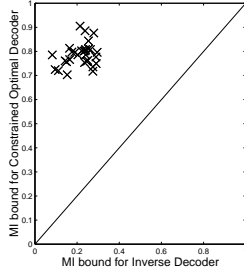

Figure 4: The bound given by the decoder $W \propto G^{-1}\mathbf{r}$ plotted against the optimised bound (for the same $G$) found using 50 updates of conjugate gradients. This was repeated over several trials of randomly chosen matrices $G$, each of which are square of $N = 10$ dimensions. For clarity, a small number of poor results (in which the bound is negative) have been omitted. To generate $G$, form the matrix $A_{ij} \sim N(0,1)$, and $B = A + A^T$. From the eigen-decomposition of $B$, i.e $BE = E\Lambda$, form $[G]_{ij} = [E\Lambda]_{ij} + 0.1N(0,1)$ (so that $G^T G$ has small off diagonal elements).

a sum of one dimensional integrals, each of which can be evaluated numerically. In the case of an orthogonal matrix $G^T G = D$ the decoding function is optimal and the MI bound is exact with the parameters in (12) set to

$$a_i = -[G^T G]_{ii}/(2\sigma^2) \qquad W = G^T/\sigma^2 \qquad b_i = [G^T G]_{ii}/\sigma^2.$$

**Optimising the linear decoder**

In the case that $G^T G$ is non-diagonal, what is the optimal linear decoder? A partial answer is given by numerically optimising the bound from (11). For the constrained case, $WG = diag(\mathbf{b})$, (12) can be used to calculate the bound. Using $W = diag(\mathbf{b})G^{-1}$,

$$\sigma^2 \mathbf{w}_i^T \mathbf{w}_i = \sigma^2 b_i^2 \sum_j ([G^{-1}]_{ij})^2,$$

and the bound depends only on $\mathbf{a}$ and $\mathbf{b}$. Under this constraint the bound can be numerically optimised as a function of $\mathbf{a}$ and $\mathbf{b}$, given a fixed vector $\sum_j ([G^{-1}]_{ij})^2$. As an alternative we can employ the decorrelation decoder, $W = G^{-1}/\sigma^2$, with $a_i = -1/(2\sigma^2)$. In fig(4) we see that, according to our bound, the decorrelation or ('inverse') decoder is suboptimal versus the linear decoder $f_i(\mathbf{r}) = a_i + \mathbf{w}_i^T \mathbf{r}$ with $W = diag(\mathbf{b})G^{-1}$, optimised over $\mathbf{a}$ and $\mathbf{b}$. These initial results are encouraging, and motivate further investigations, for example, using syndrome decoding for CDMA.

## 6  Posterior Approximations

There is an interesting relationship between maximising the bound on the MI and computing an optimal estimate $q(\mathbf{s}|\mathbf{r})$ of an intractable posterior $p(\mathbf{s}|\mathbf{r})$. The optimal bit error solution sets $q(s_i|\mathbf{r})$ to the mean of the exact posterior marginal $p(s_i|\mathbf{r})$. Mean Field Theory approximates the posterior marginal by minimising the KL divergence: $KL(q||p) = \sum_{\mathbf{s}} (q(\mathbf{s}|\mathbf{r}) \log q(\mathbf{s}|\mathbf{r}) - q(\mathbf{s}|\mathbf{r}) \log p(\mathbf{s}|\mathbf{r}))$, where $q(\mathbf{s}|\mathbf{r}) = \prod_i q(s_i|\mathbf{r})$. In this case, the KL divergence is tractably computable (up to a neglectable prefactor). However, this form of the KL divergence chooses $q(s_i|\mathbf{r})$ to be any one of a very large number of local modes of the posterior distribution $p(s_i|\mathbf{r})$. Since the optimal choice is to choose the posterior marginal *mean*, this is why using Mean Field decoding is generally suboptimal. Alternatively, consider

$$KL(p||q) = \sum_{\mathbf{s}} (p(\mathbf{s}|\mathbf{r}) \log p(\mathbf{s}|\mathbf{r}) - p(\mathbf{s}|\mathbf{r}) \log q(\mathbf{s}|\mathbf{r})) = -\sum_{\mathbf{s}} p(\mathbf{s}|\mathbf{r}) \log q(\mathbf{s}|\mathbf{r}) + const.$$

This is the correct KL divergence in the sense that, optimally, $q(s_i|\mathbf{r}) = p(s_i|\mathbf{r})$, that is, the posterior marginal is correctly calculated. The difficulty lies in performing

averages with respect to $p(\mathbf{s}|\mathbf{r})$, which are generally intractable. Since we will have a distribution $p(\mathbf{r})$ it is reasonable to provide an averaged objective function,

$$\sum_{\mathbf{r}}\sum_{\mathbf{s}} p(\mathbf{r})p(\mathbf{s}|\mathbf{r})\log q(\mathbf{s}|\mathbf{r}) = \sum_{\mathbf{r}}\sum_{\mathbf{s}} p(\mathbf{s})p(\mathbf{r}|\mathbf{s})\log q(\mathbf{s}|\mathbf{r}). \qquad (13)$$

Whilst, for any given $\mathbf{r}$, we cannot calculate the best posterior marginal estimate, we may be able to calculate the best posterior marginal estimate *on average*. This is precisely the case in, for example, CDMA since the average over $p(\mathbf{r}|\mathbf{s})$ is tractable, and the resulting average over $p(\mathbf{s})$ can be well approximated numerically. Wherever an average case objective is desired is of interest to the methods suggested here.

## 7  Discussion

We have described a general theoretically justified approach to information maximization in noisy channels. Whilst the bound is straightforward, it appears to have attracted little previous attention as a practical tool for MI optimisation. We have shown how it naturally generalises linear compression and feature extraction. It is a more direct approach to optimal coding than using the Fisher 'Information' in neurobiological population encoding. Our bound enables a principled comparison of different information maximisation algorithms, and may have applications in other areas of machine learning and Information Theory, such as error-correction.

[1] D. Barber, *Tractable Approximate Belief Propagation*, Advanced Mean Field Methods Theory and Practice (D. Saad and M. Opper, eds.), MIT Press, 2001.

[2] H. Barlow, *Unsupervised Learning*, Neural Computation **1** (1989), 295–311.

[3] S Becker, *An Information-theoretic unsupervised learning algorithm for neural networks*, Ph.D. thesis, University of Toronto, 1992.

[4] A.J. Bell and T.J. Sejnowski, *An information-maximisation approach to blind separation and blind deconvolution*, Neural Computation **7** (1995), no. 6, 1004–1034.

[5] N. Brunel and J.-P. Nadal, *Mutual Information, Fisher Information and Population Coding*, Neural Computation **10** (1998), 1731–1757.

[6] T. Jaakkola and M. Jordan., *Improving the mean field approximation via the use of mixture distributions*, Proceedings of the NATO ASI on Learning in Graphical Models, Kluwer, 1997.

[7] R. Linsker, *Deriving Receptive Fields Using an Optimal Encoding Criterion*, Advances in Neural Information Processing Systems (Lee Giles (eds) Steven Hanson, Jack Cowan, ed.), vol. 5, Morgan-Kaufmann, 1993.

[8] S. Mika, B. Schoelkopf, A.J. Smola, K-R. Muller, M. Scholz, and Gunnar Ratsch, *Kernel PCA and De-Noising in Feature Spaces*, Advances in Neural Information Processing Systems **11** (1999).

[9] R. M. Neal and G. E. Hinton, *A View of the EM Algorithm That Justifies Incremental, Sparse, and Other Variants*, Learning in Graphical Models (M.J. Jordan, ed.), MIT Press, 1998.

[10] D. Saad and M. Opper, *Advanced Mean Field Methods Theory and Practice*, MIT Press, 2001.

[11] T. Tanaka, *Analysis of Bit Error Probability of Direct-Sequence CDMA Multiuser Demodulators*, Advances in Neural Information Processing Systems (T. K. Leen et al. (eds.), ed.), vol. 13, MIT Press, 2001, pp. 315–321.

[12] K. Torkkola and W. M. Campbell, *Mutual Information in Learning Feature Transformations*, Proc. 17th International Conf. on Machine Learning (2000).

[13] M. Wainwright, T. Jaakkola, and A. Willsky, *A new class of upper bounds on the log partition function*, Uncertainty in Artificial Intelligence, 2002.

## Footnotes

[1]Other variational methods may be considered to approximate the normalisation constant of $p(\mathbf{s}|\mathbf{r})$[13], and it would be interesting to look into the possibility of using them in a MI approximation, and also as approximate decoding algorithms.
